# Layered Image Motion with Explicit Occlusions, Temporal Consistency, and Depth Ordering

**Deqing Sun, Erik B. Sudderth, and Michael J. Black**
Department of Computer Science, Brown University
{dqsun,sudderth,black}@cs.brown.edu

## Abstract

Layered models are a powerful way of describing natural scenes containing smooth surfaces that may overlap and occlude each other. For image motion estimation, such models have a long history but have not achieved the wide use or accuracy of non-layered methods. We present a new probabilistic model of optical flow in layers that addresses many of the shortcomings of previous approaches. In particular, we define a probabilistic graphical model that explicitly captures: 1) occlusions and disocclusions; 2) depth ordering of the layers; 3) temporal consistency of the layer segmentation. Additionally the optical flow in each layer is modeled by a combination of a parametric model and a smooth deviation based on an MRF with a robust spatial prior; the resulting model allows *roughness in layers*. Finally, a key contribution is the formulation of the layers using an image-dependent hidden field prior based on recent models for static scene segmentation. The method achieves state-of-the-art results on the Middlebury benchmark and produces meaningful scene segmentations as well as detected occlusion regions.

## 1  Introduction

Layered models of scenes offer significant benefits for optical flow estimation [8, 11, 25]. Splitting the scene into layers enables the motion in each layer to be defined more simply, and the estimation of motion boundaries to be separated from the problem of smooth flow estimation. Layered models also make reasoning about occlusion relationships easier. In practice, however, none of the current top performing optical flow methods use a layered approach [2]. The most accurate approaches are single-layered, and instead use some form of robust smoothness assumption to cope with flow discontinuities [5]. This paper formulates a new probabilistic, layered motion model that addresses the key problems of previous layered approaches. At the time of writing, it achieves the lowest average error of all tested approaches on the Middlebury optical flow benchmark [2]. In particular, the accuracy at occlusion boundaries is significantly better than previous methods. By segmenting the observed scene, our model also identifies occluded and disoccluded regions.

Layered models provide a segmentation of the scene and this segmentation, because it corresponds to scene structure, should persist over time. However, this persistence is not a benefit if one is only computing flow between two frames; this is one reason that multi-layer models have not surpassed their single-layer competitors on two-frame benchmarks. Without loss of generality, here we use three-frame sequences to illustrate our method. In practice, these three frames can be constructed from an image pair by computing both the forward and backward flow. The key is that this gives two segmentations of the scene, one at each time instant, both of which must be consistent with the flow. We formulate this *temporal layer consistency* probabilistically. Note that the assumption of temporal layer consistency is much more realistic than previous assumptions of temporal motion consistency [4]; while the scene motion can change rapidly, scene structure persists.

One of the main motivations for layered models is that, conditioned on the segmentation into layers, each layer can employ affine, planar, or other strong models of optical flow. By applying a single smooth motion across the entire layer, these models combine information over long distances and interpolate behind occlusions. Such rigid parametric assumptions, however, are too restrictive for real scenes. Instead one can model the flow within each layer as smoothly varying [26]. While the resulting model is more flexible than traditional parametric models, we find that it is still not as accurate as robust single-layer models. Consequently, we formulate a hybrid model that combines a base affine motion with a robust Markov random field (MRF) model of *deformations* from affine [6]. This *roughness in layers* model, which is similar in spirit to work on plane+parallax [10, 14, 19], encourages smooth flow within layers but allows significant local deviations.

Because layers are temporally persistent, it is also possible to reason about their relative depth ordering. In general, reliable recovery of depth order requires three or more frames. Our probabilistic formulation explicitly orders layers by depth, and we show that the correct order typically produces more probable (lower energy) solutions. This also allows explicit reasoning about occlusions, which our model predicts at locations where the layer segmentations for consecutive frames disagree.

Many previous layered approaches are not truly "layered": while they segment the image into multiple regions with distinct motions, they do not model what is in front of what. For example, widely used MRF models [27] encourage neighboring pixels to occupy the same region, but do not capture relationships between regions. In contrast, building on recent state-of-the-art results in static scene segmentation [21], our model determines layer support via an ordered sequence of occluding binary masks. These binary masks are generated by thresholding a series of random, continuous functions. This approach uses image-dependent Gaussian random field priors and favors partitions which accurately match the statistics of real scenes [21]. Moreover, the continuous layer support functions play a key role in accurately modeling temporal layer consistency. The resulting model produces accurate layer segmentations that improve flow accuracy at occlusion boundaries, and recover meaningful scene structure.

As summarized in Figure 1, our method is based on a principled, probabilistic generative model for image sequences. By combining recent advances in dense flow estimation and natural image segmentation, we develop an algorithm that simultaneously estimates accurate flow fields, detects occlusions and disocclusions, and recovers the layered structure of realistic scenes.

## 2   Previous Work

Layered approaches to motion estimation have long been seen as elegant and promising, since spatial smoothness is separated from the modeling of discontinuities and occlusions. Darrell and Pentland [7, 8] provide the first full approach that incorporates a Bayesian model, "support maps" for segmentation, and robust statistics. Wang and Adelson [25] clearly motivate layered models of image sequences, while Jepson and Black [11] formalize the problem using probabilistic mixture models. A full review of more recent methods is beyond our scope [1, 3, 12, 13, 16, 17, 20, 24, 27, 29].

Early methods, which use simple parametric models of image motion within layers, are not highly accurate. Observing that rigid parametric models are too restrictive for real scenes, Weiss [26] uses a more flexible Gaussian process to describe the motion within each layer. Even using modern implementation methods [22] this approach does not achieve state-of-the-art results. Allocating a separate layer for every small surface discontinuity is impractical and fails to capture important global scene structure. Our approach, which allows "roughness" within layers rather than "smoothness," provides a compromise that captures coarse scene structure as well as fine within-layer details.

One key advantage of layered models is their ability to realistically model occlusion boundaries. To do this properly, however, one must know the relative depth order of the surfaces. Performing inference over the combinatorial range of possible occlusion relationships is challenging and, consequently, only a few layered flow models explicitly encode relative depth [12, 30]. Recent work revisits the layered model to handle occlusions [9], but does not explicitly model the layer ordering or achieve state-of-the-art performance on the Middlebury benchmark. While most current optical flow methods are "two-frame," layered methods naturally extend to longer sequences [12, 29, 30].

Layered models all have some way of making either a hard or soft assignment of pixels to layers. Weiss and Adelson [27] introduce spatial coherence to these layer assignments using a spatial MRF

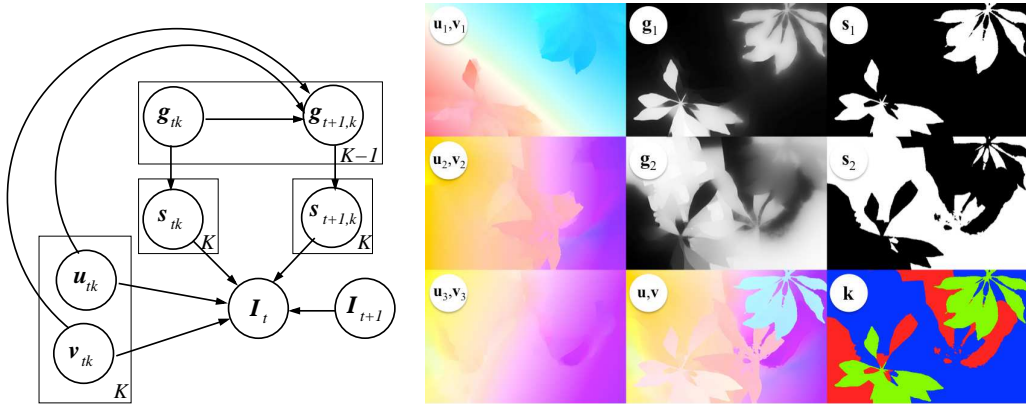

Figure 1: *Left:* Graphical representation for the proposed layered model. *Right:* Illustration of variables from the graphical model for the "Schefflera" sequence. Labeled sub-images correspond to nodes in the graph. The left column shows the flow fields for three layers, color coded as in [2]. The $\mathbf{g}$ and $\mathbf{s}$ images illustrate the reasoning about layer ownership (see text). The composite flow field $(\mathbf{u}, \mathbf{v})$ and layer labels $(\mathbf{k})$ are also shown.

model. However, the Ising/Potts MRF they employ assigns low probability to typical segmentations of natural scenes [15]. Adapting recent work on static image segmentation by Sudderth and Jordan [21], we instead generate spatially coherent, ordered layers by thresholding a series of random continuous functions. As in the single-image case, this approach realistically models the size and shape properties of real scenes. For motion estimation there are additional advantages: it allows accurate reasoning about occlusion relationships and modeling of temporal layer consistency.

## 3 A Layered Motion Model

Building on this long sequence of prior work, our generative model of layered image motion is summarized in Figure 1. Below we describe how the generative model captures piecewise smooth deviation of the layer motion from parametric models (Sec. 3.1), depth ordering and temporal consistency of layers (Sec. 3.2), and regions of occlusion and disocclusion (Sec. 3.3).

### 3.1 Roughness in Layers

Our approach is inspired by Weiss's model of smoothness in layers [26]. Given a sequence of images $\mathbf{I}_t, 1 \leq t \leq T$, we model the evolution from the current frame $\mathbf{I}_t$, to the subsequent frame $\mathbf{I}_{t+1}$, via $K$ locally smooth, but potentially globally complex, flow fields. Let $\mathbf{u}_{tk}$ and $\mathbf{v}_{tk}$ denote the horizontal and vertical flow fields, respectively, for layer $k$ at time $t$. The corresponding flow vector for pixel $(i, j)$ is then denoted by $(u_{tk}^{ij}, v_{tk}^{ij})$.

Each layer's flow field is drawn from a distribution chosen to encourage piecewise smooth motion. For example, a pairwise Markov random field (MRF) would model the horizontal flow field as

$$p(\mathbf{u}_{tk}) \propto \exp\{-E_{\mathrm{mrf}}(\mathbf{u}_{tk})\} = \exp\left\{ -\frac{1}{2} \sum_{(i,j)} \sum_{(i',j') \in \Gamma(i,j)} \rho_s(u_{tk}^{ij} - u_{tk}^{i'j'}) \right\}. \tag{1}$$

Here, $\Gamma(i, j)$ is the set of neighbors of pixel $(i, j)$, often its four nearest neighbors. The potential $\rho_s(\cdot)$ is some robust function [5] that encourages smoothness, but allows occasional significant deviations from it. The vertical flow field $\mathbf{v}_{tk}$ can then be modeled via an independent MRF prior as in Eq. (1), as justified by the statistics of natural flow fields [18].

While such MRF priors are flexible, they capture very little dependence between pixels separated by even moderate image distances. In contrast, real scenes exhibit coherent motion over large scales, due to the motion of (partially) rigid objects in the world. To capture this, we associate an affine (or planar) motion model, with parameters $\theta_{tk}$, to each layer $k$. We then use an MRF to allow piecewise smooth *deformations* from the globally rigid assumptions of affine motion:

$$E_{\mathrm{aff}}(\mathbf{u}_{tk}, \theta_{tk}) = \frac{1}{2} \sum_{(i,j)} \sum_{(i',j') \in \Gamma(i,j)} \rho_s\left( (u_{tk}^{ij} - \bar{u}_{\theta_{tk}}^{ij}) - (u_{tk}^{i'j'} - \bar{u}_{\theta_{tk}}^{i'j'}) \right). \tag{2}$$

Here, $\bar{u}_{\theta_{tk}}^{ij}$ denotes the horizontal motion predicted for pixel $(i, j)$ by an affine model with parameters $\theta_{tk}$. Unlike classical models that assume layers are globally well fit by a single affine motion [6, 25], this prior allows significant, locally smooth deviations from rigidity. Unlike the basic smoothness prior of Eq. (1), this semiparametric construction allows effective global reasoning about non-contiguous segments of partially occluded objects. More sophisticated flow deformation priors may also be used, such as those based on robust non-local terms [22, 28].

## 3.2 Layer Support and Spatial Contiguity

The support for whether or not a pixel belongs to a given layer $k$ is defined using a hidden random field $\mathbf{g}_k$. We associate each of the first $K - 1$ layers at time $t$ with a random continuous function $\mathbf{g}_{tk}$, defined over the same domain as the image. This hidden support field is illustrated in Figure 1.

We assume a single, unique layer is observable at each location and that the observed motion of that pixel is determined by its assigned layer. Analogous to level set representations, the discrete support of each layer is determined by thresholding $\mathbf{g}_{tk}$: pixel $(i, j)$ is considered visible when $g_{tk}(i, j) \geq 0$. Let $s_{tk}(i, j)$ equal one if layer $k$ is visible at pixel $(i, j)$, and zero otherwise; note that $\sum_k s_{tk}(i, j) = 1$. For pixels $(i, j)$ for which $g_{tk}(i, j) < 0$, we necessarily have $s_{tk}(i, j) = 0$.

We define the layers to be ordered with respect to the camera, so that layer $k$ occludes layers $k' > k$. Given the full set of support functions $\mathbf{g}_{tk}$, the unique layer $k_{t*}^{ij}$ for which $s_{tk_{t*}^{ij}}(i, j) = 1$ is then

$$k_{t*}^{ij} = \min\left(\{k \mid 1 \leq k \leq K - 1, g_{tk}(i, j) \geq 0\} \cup \{K\}\right). \tag{3}$$

Note that layer $K$ is essentially a background layer that captures all pixels not assigned to the first $K - 1$ layers. For this reason, only $K - 1$ hidden fields $\mathbf{g}_{tk}$ are needed (see Figure 1).

Our use of thresholded, random continuous functions to define layer support is partially motivated by known shortcomings of discrete Ising/Potts MRF models for image partitions [15]. They also provide a convenient framework for modeling the temporal and spatial coherence observed in real motion sequences. Spatial coherence is captured via a Gaussian conditional random field in which edge weights are modulated by local differences in Lab color vectors, $\mathbf{I}_t^c(i, j)$:

$$E_{\text{space}}(\mathbf{g}_{tk}) = \frac{1}{2} \sum_{(i,j)} \sum_{(i',j') \in \Gamma(i,j)} w_{i'j'}^{ij} (g_{tk}(i, j) - g_{tk}(i', j'))^2, \tag{4}$$

$$w_{i'j'}^{ij} = \max\left\{ \exp\left\{ -\frac{1}{2\sigma_c^2} ||\mathbf{I}_t^c(i, j) - \mathbf{I}_t^c(i', j')||^2 \right\}, \delta_c \right\}. \tag{5}$$

The threshold $\delta_c > 0$ adds robustness to large color changes in internal object texture. Temporal coherence of surfaces is then encouraged via a corresponding Gaussian MRF:

$$E_{\text{time}}(\mathbf{g}_{tk}, \mathbf{g}_{t+1,k}, \mathbf{u}_{tk}, \mathbf{v}_{tk}) = \sum_{(i,j)} (g_{tk}(i, j) - g_{t+1,k}(i + u_{tk}^{ij}, j + v_{tk}^{ij}))^2. \tag{6}$$

Critically, this energy function uses the corresponding flow field to non-rigidly align the layers at subsequent frames. By allowing smooth deformation of the support functions $\mathbf{g}_{tk}$, we allow layer support to evolve over time, as opposed to transforming a single rigid template [12].

Our model of layer coherence is inspired by a recent method for image segmentation, based on spatially dependent Pitman-Yor processes [21]. That work makes connections between layered occlusion processes and *stick breaking* representations of nonparametric Bayesian models. By assigning appropriate stochastic priors to layer thresholds, the Pitman-Yor model captures the power law statistics of natural scene partitions and infers an appropriate number of segments for each image. Existing optical flow benchmarks employ artificially constructed scenes that may have different layer-level statistics. Consequently our experiments in this paper employ a fixed number of layers $K$.

## 3.3 Depth Ordering and Occlusion Reasoning

The preceding generative process defines a set of $K$ ordered layers, with corresponding flow fields $\mathbf{u}_{tk}, \mathbf{v}_{tk}$ and segmentation masks $\mathbf{s}_{tk}$. Recall that the layer assignment masks $\mathbf{s}$ are a

deterministic function (threshold) of the underlying continuous layer support functions $\mathbf{g}$ (see Eq. (3)). To consistently reason about occlusions, we examine the layer assignments $s_{tk}(i,j)$ and $s_{t+1,k}(i+u_{tk}^{ij}, j+v_{tk}^{ij})$ at locations corresponded by the underlying flow fields. This leads to a far richer occlusion model than standard spatially independent outlier processes: geometric consistency is enforced via the layered sequence of flow fields.

Let $\mathbf{I}_t^s(i,j)$ denote an observed image feature for pixel $(i,j)$; we work with a filtered version of the intensity images to provide some invariance to illumination changes. If $s_{tk}(i,j) = s_{t+1,k}(i+u_{tk}^{ij}, j+v_{tk}^{ij}) = 1$, the visible layer for pixel $(i,j)$ at time $t$ remains unoccluded at time $t+1$, and the image observations are modeled using a standard brightness (or, here, feature) constancy assumption. Otherwise, that pixel has become occluded, and is instead generated from a uniform distribution. The image likelihood model can then be written as

$$p(\mathbf{I}_t^s \mid \mathbf{I}_{t+1}^s, \mathbf{u}_t, \mathbf{v}_t, \mathbf{g}_t, \mathbf{g}_{t+1}) \propto \exp\{-E_{\text{data}}(\mathbf{u}_t, \mathbf{v}_t, \mathbf{g}_t, \mathbf{g}_{t+1})\}$$

$$= \exp\Big\{ -\sum_k \sum_{(i,j)} \Big(\rho_d(\mathbf{I}_t^s(i,j) - \mathbf{I}_{t+1}^s(i+u_{tk}^{ij}, j+v_{tk}^{ij}))s_{tk}(i,j)s_{t+1,k}(i+u_{tk}^{ij}, j+v_{tk}^{ij})$$

$$+ \lambda_d s_{tk}(i,j)(1 - s_{t+1,k}(i+u_{tk}^{ij}, j+v_{tk}^{ij}))\Big)\Big\}$$

where $\rho_d(\cdot)$ is a robust potential function and the constant $\lambda_d$ arises from the difference of the log normalization constants for the robust and uniform distributions. With algebraic simplifications, the data error term can be written as

$$E_{\text{data}}(\mathbf{u}_t, \mathbf{v}_t, \mathbf{g}_t, \mathbf{g}_{t+1}) =$$

$$\sum_k \sum_{(i,j)} \Big(\rho_d(\mathbf{I}_t^s(i,j) - \mathbf{I}_{t+1}^s(i+u_{tk}^{ij}, j+v_{tk}^{ij})) - \lambda_d\Big)s_{tk}(i,j)s_{t+1,k}(i+u_{tk}^{ij}, j+v_{tk}^{ij}) \quad (7)$$

up to an additive, constant multiple of $\lambda_d$. The shifted potential function $(\rho_d(\cdot) - \lambda_d)$ represents the change in energy when a pixel transitions from an occluded to an unoccluded configuration. Note that occlusions have higher likelihood only for sufficiently large discrepancies in matched image features and can only occur via a corresponding change in layer visibility.

## 4 Posterior Inference from Image Sequences

Considering the full generative model defined in Sec. 3, *maximum a posteriori* (MAP) estimation for a $T$ frame image sequence is equivalent to minimization of the following energy function:

$$E(\mathbf{u}, \mathbf{v}, \mathbf{g}, \theta) = \sum_{t=1}^{T-1} \Big\{ E_{\text{data}}(\mathbf{u}_t, \mathbf{v}_t, \mathbf{g}_t, \mathbf{g}_{t+1}) + \sum_{k=1}^{K} \lambda_a (E_{\text{aff}}(\mathbf{u}_{tk}, \theta_{tk}) + E_{\text{aff}}(\mathbf{v}_{tk}, \theta_{tk}))$$

$$+ \sum_{k=1}^{K-1} \lambda_b E_{\text{space}}(\mathbf{g}_{tk}) + \lambda_c E_{\text{time}}(\mathbf{g}_{tk}, \mathbf{g}_{t+1,k}, \mathbf{u}_{tk}, \mathbf{v}_{tk}) \Big\} + \sum_{k=1}^{K-1} \lambda_b E_{\text{space}}(\mathbf{g}_{Tk}). \quad (8)$$

Here $\lambda_a$, $\lambda_b$, and $\lambda_c$ are weights controlling the relative importance of the affine, spatial, and temporal terms respectively. Simultaneously inferring flow fields, layer support maps, and depth ordering is a challenging process; our approach is summarized below.

### 4.1 Relaxation of the Layer Assignment Process

Due to the non-differentiability of the threshold process that determines assignments of regions to layers, direct minimization of Eq. (8) is challenging. For a related approach to image segmentation, a mean field variational method has been proposed [21]. However, that segmentation model is based on a much simpler, spatially factorized likelihood model for color and texture histogram features. Generalization to the richer flow likelihoods considered here raises significant complications.

Instead, we relax the hard threshold assignment process using the logistic function $\sigma(g) = 1/(1 + \exp(-g))$. Applied to Eq. (3), this induces the following soft layer assignments:

$$\tilde{s}_{tk}(i,j) = \begin{cases} \sigma(\lambda_e g_{tk}(i,j)) \prod_{k'=1}^{k-1} \sigma(-\lambda_e g_{tk'}(i,j)), & 1 \leq k < K, \\ \prod_{k'=1}^{K-1} \sigma(-\lambda_e g_{tk'}(i,j)), & k = K. \end{cases} \quad (9)$$

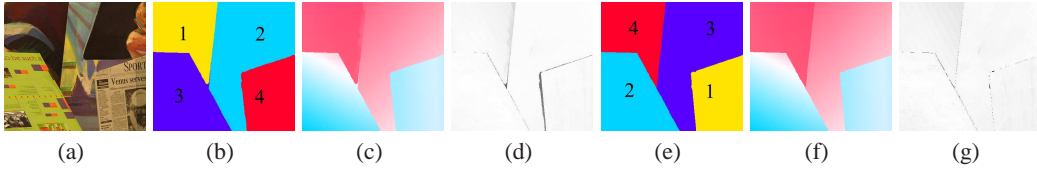

| (a) | (b) | (c) | (d) | (e) | (f) | (g) |

Figure 2: Results on the "Venus" sequence with 4 layers. The two background layers move faster than the two foreground layers, and the solution with the correct depth ordering has lower energy and smaller error. (a) First frame. (b-d) Fast-to-slow ordering: EPE $0.252$ and energy $-1.786 \times 10^6$. Left to right: motion segmentation, estimated flow field, and absolute error of estimated flow field. (f-g) Slow-to-fast ordering: EPE $0.195$ and energy $-1.808 \times 10^6$. Darker indicates larger flow field errors in (d) and (g).

Note that $\sigma(-g) = 1 - \sigma(g)$, and $\sum_{k=1}^{K} \tilde{s}_{tk}(i,j) = 1$ for any $g_{tk}$ and constant $\lambda_e > 0$.

Substituting these soft assignments $\tilde{s}_{tk}(i,j)$ for $s_{tk}(i,j)$ in Eq. (7), we obtain a differentiable energy function that can be optimized via gradient-based methods. A related relaxation underlies the classic backpropagation algorithm for neural network training.

### 4.2  Gradient-Based Energy Minimization

We estimate the hidden fields for all the frames together, while fixing the flow fields, by optimizing an objective involving the relevant $E_{\text{data}}(\cdot)$, $E_{\text{space}}(\cdot)$, and $E_{\text{time}}(\cdot)$ terms. We then estimate the flow fields $\mathbf{u}_t, \mathbf{v}_t$ for each frame, while fixing those of neighboring frames and the hidden fields, via the $E_{\text{data}}(\cdot)$, $E_{\text{aff}}(\cdot)$, and $E_{\text{time}}(\cdot)$ terms. For flow estimation, we use a standard coarse-to-fine, warping-based technique as described in [22]. For hidden field estimation, we use an implementation of conjugate gradient descent with backtracking and line search. See *Supplemental Material* for details.

## 5  Experimental Results

We apply the proposed model to two-frame sequences and compute both the forward and backward flow fields. This enables the use of the temporal consistency term by treating one frame as both the previous and the next frame of the other[1]. We obtain an initial flow field using the Classic+NL method [22], cluster the flow vectors into $K$ groups (layers), and convert the initial segmentation into the corresponding hidden fields. We then use a two-level Gaussian pyramid (downsampling factor $0.8$) and perform a fairly standard incremental estimation of the flow fields for each layer. At each level, we perform $20$ incremental warping steps and during each step alternately solve for the hidden fields and the flow estimates. In the end, we threshold the hidden fields to compute a hard segmentation, and obtain the final flow field by selecting the flow field from the appropriate layers.

Occluded regions are determined by inconsistencies between the hard segmentations at subsequent frames, as matched by the final flow field. We would ideally like to compare layer initializations based on all permutations of the initial flow vector clusters, but this would be computationally intensive for large $K$. Instead we compare two orders: a fast-to-slow order appropriate for rigid scenes, and an opposite slow-to-fast order (for variety and robustness). We illustrate automatic selection of the preferred order for the "Venus" sequence in Figure 2.

The parameters for all experiments are set to $\lambda_a = 3$, $\lambda_b = 30$, $\lambda_c = 4$, $\lambda_d = 9$, $\lambda_e = 2$, $\sigma_i = 12$, and $\delta_c = 0.004$. A generalized Charbonnier function is used for $\rho_S(\cdot)$ and $\rho_d(\cdot)$ (see *Supplemental Material*). Optimization takes about $5$ hours for the two-frame "Urban" sequence using our MATLAB implementation.

### 5.1  Results on the Middlebury Benchmark

**Training Set**  As a baseline, we implement the smoothness in layers model [26] using modern techniques, and obtain an average training end-point error (EPE) of $0.487$. This is reasonable but not competitive with state-of-the-art methods. The proposed model with $1$ to $4$ layers produces average EPEs of $0.248$, $0.212$, $0.200$, and $0.194$, respectively (see Table 1). The one-layer model is similar to the Classic+NL method, but has a less sophisticated (more local) model of the flow within

Table 1: Average end-point error (EPE) on the Middlebury optical flow benchmark *training* set.

| | Avg. EPE | Venus | Dimetrodon | Hydrangea | RubberWhale | Grove2 | Grove3 | Urban2 | Urban3 |
|---|---|---|---|---|---|---|---|---|---|
| **Weiss [26]** | 0.487 | 0.510 | 0.179 | 0.249 | 0.236 | 0.221 | 0.608 | 0.614 | 1.276 |
| **Classic++** | 0.285 | 0.271 | 0.128 | 0.153 | 0.081 | 0.139 | 0.614 | 0.336 | 0.555 |
| **Classic+NL** | 0.221 | 0.238 | 0.131 | 0.152 | 0.073 | 0.103 | 0.468 | 0.220 | 0.384 |
| **1layer** | 0.248 | 0.243 | 0.144 | 0.175 | 0.095 | 0.125 | 0.504 | 0.279 | 0.422 |
| **2layers** | 0.212 | 0.219 | 0.147 | 0.169 | 0.081 | 0.098 | 0.376 | 0.236 | 0.370 |
| **3layers** | 0.200 | 0.212 | 0.149 | 0.173 | 0.073 | 0.090 | 0.343 | 0.220 | 0.338 |
| **4layers** | 0.194 | 0.197 | 0.148 | 0.159 | 0.068 | 0.088 | 0.359 | 0.230 | 0.300 |
| **3layers w/ WMF** | 0.195 | 0.211 | 0.150 | 0.161 | 0.067 | 0.086 | 0.331 | 0.210 | 0.345 |
| **3layers w/ WMF C++Init** | 0.203 | 0.212 | 0.151 | 0.161 | 0.066 | 0.087 | 0.339 | 0.210 | 0.396 |

Table 2: Average end-point error (EPE) on the Middlebury optical flow benchmark *test set*.

| | Rank | Average | Army | Mequon | Schefflera | Wooden | Grove | Urban | Yosemite | Teddy |
|---|---|---|---|---|---|---|---|---|---|---|
| EPE | | | | | | | | | | |
| **Layers++** | 4.3 | 0.270 | 0.08 | 0.19 | 0.20 | 0.13 | 0.48 | 0.47 | 0.15 | 0.46 |
| **Classic+NL** | 6.5 | 0.319 | 0.08 | 0.22 | 0.29 | 0.15 | 0.64 | 0.52 | 0.16 | 0.49 |
| EPE in boundary regions | | | | | | | | | | |
| **Layers++** | | 0.560 | 0.21 | 0.56 | 0.40 | 0.58 | 0.70 | 1.01 | 0.14 | 0.88 |
| **Classic+NL** | | 0.689 | 0.23 | 0.74 | 0.65 | 0.73 | 0.93 | 1.12 | 0.13 | 0.98 |

that layer. It thus performs worse than the Classic+NL initialization; the performance improvements allowed by additional layers demonstrate the benefits of a layered model.

Accuracy is improved by applying a $15 \times 15$ *weighted median filter* (WMF) [22] to the flow fields of each layer during the iterative warping step (EPE for $1$ to $4$ layers: $0.231$, $0.204$, $0.195$, and $0.193$). Weighted median filtering can be interpreted as a non-local spatial smoothness term in the energy function that integrates flow field information over a larger spatial neighborhood.

The "correct" number of layers for a real scene is not well defined (consider the "Grove3" sequence, for example). We use a restricted number of layers, and model the remaining complexity of the flow within each layer via the roughness-in-layers spatial term and the WMF. As the number of layers increases, the complexity of the flow within each layer decreases, and consequently the need for WMF also decreases; note that the difference in EPE for the 4-layer model with and without WMF is insignificant. For the remaining experiments we use the version with WMF.

To test the sensitivity of the result to the initialization, we also initialized with Classic++ ("C++Init" in Table 1), a good, but not top, non-layered method [22]. The average EPE for $1$ to $4$ layers increases to $0.248$, $0.206$, $0.203$, and $0.198$, respectively. While the one-layer method gets stuck in poor local minima on the "Grove3" and "Urban3" sequences, models with additional layers are more robust to the initialization. For more details and full EPE results, see the *Supplemental Material*.

**Test Set** For evaluation, we focus on a model with 3 layers (denoted "Layers++" in the Middlebury public table). On the Middlebury test set it has an average EPE of $0.270$ and average angular error (AAE) of $2.556$; this is the lowest among all tested methods [2] at the time of writing (Oct. 2010). Table 2 summarizes the results for individual test sequences. The layered model is particularly accurate at motion boundaries, probably due to the use of layer-specific motion models, and the explicit modeling of occlusion in $E_{\text{data}}$ (Eq. (7)). For more extensive results, see the *Supplemental Material*.

**Visual Comparison** Figure 3 shows results for the 3-layer model on several training and test sequences. Notice that the layered model produces a motion segmentation that captures the major structure of the scene, and the layer boundaries correspond well to static image edges. It detects most occlusion regions and interpolates their motion reasonably well. Several sequences show significant improvement due to the global reasoning provided by the layered model. On the training "Grove3" sequence, the proposed method correctly identifies many holes between the branches and leaves as background. It also associates the branch at the bottom right corner with branches in the center. As the branch moves beyond the image boundary, the layered model interpolates its motion using long-range correlation with the branches in the center. In contrast, the single-layered approach incorrectly interpolates from local background regions. The "Schefflera" result illustrates how the layered method can separate foreground objects from the background (e.g., the leaves in the top right corner), and thereby reduce errors made by single-layer approaches such as Classic+NL.

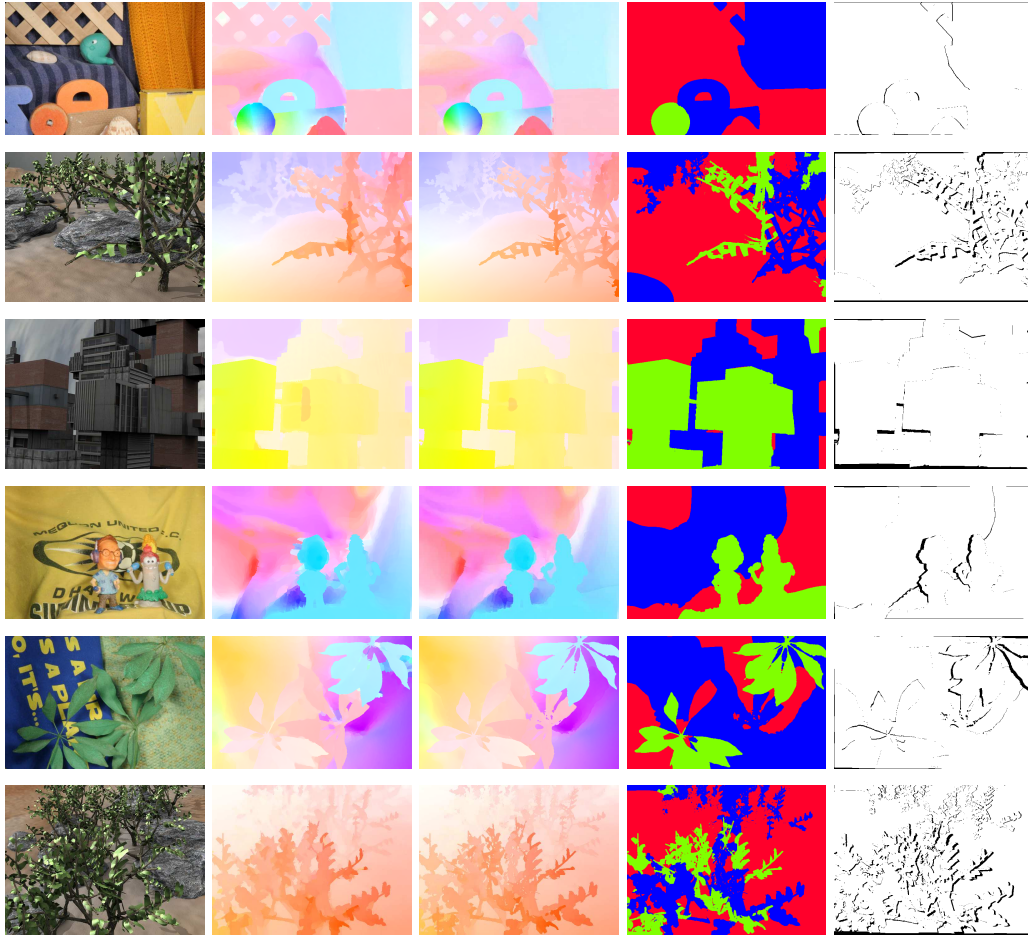

Figure 3: Results on some Middlebury training (rows 1 to 3) and test (rows 4 to 6) sequences. *Top to bottom:* "RubberWhale", "Grove3", "Urban3", "Mequon", "Schefflera", and "Grove". *Left to right:* First image frame, initial flow field from "Classic+NL", final flow field, motion segmentation (green front, blue middle, red back), and detected occlusions. Best viewed in color and enlarged to allow comparison of detailed motions.

## 6 Conclusion and Discussion

We have described a new probabilistic formulation for layered image motion that explicitly models occlusion and disocclusion, depth ordering of layers, and the temporal consistency of the layer segmentation. The approach allows the flow field in each layer to have piecewise smooth deformation from a parametric motion model. Layer support is modeled using an image-dependent hidden field prior that supports a model of temporal layer continuity over time. The image data error term takes into account layer occlusion relationships, resulting in increased flow accuracy near motion boundaries. Our method achieves state-of-the-art results on the Middlebury optical flow benchmark while producing meaningful segmentation and occlusion detection results.

Future work will address better inference methods, especially a better scheme to infer the layer order, and the automatic estimation of the number of layers. Computational efficiency has not been addressed, but will be important for inference on long sequences. Currently our method does not capture transparency, but this could be supported using a soft layer assignment and a different generative model. Additionally, the parameters of the model could be learned [23], but this may require more extensive and representative training sets. Finally, the parameters of the model, especially the number of layers, should adapt to the motions in a given sequence.

**Acknowledgments**   DS and MJB were supported in part by the NSF Collaborative Research in Computational Neuroscience Program (IIS–0904875) and a gift from Intel Corp.

## Footnotes

[1]Our model works for longer sequences. We use two frames here for fair comparison with other methods.

# References

[1] S. Ayer and H. S. Sawhney. Layered representation of motion video using robust maximum-likelihood estimation of mixture models and MDL encoding. In *ICCV*, pages 777–784, Jun 1995.

[2] S. Baker, D. Scharstein, J. P. Lewis, S. Roth, M. J. Black, and R. Szeliski. A database and evaluation methodology for optical flow. *IJCV*, to appear.

[3] S. Birchfield and C. Tomasi. Multiway cut for stereo and motion with slanted surfaces. In *ICCV*, pages 489–495, 1999.

[4] M. J. Black and P. Anandan. Robust dynamic motion estimation over time. In *CVPR*, pages 296–302, 1991.

[5] M. J. Black and P. Anandan. The robust estimation of multiple motions: Parametric and piecewise-smooth flow fields. *CVIU*, 63:75–104, 1996.

[6] M. J. Black and A. D. Jepson. Estimating optical-flow in segmented images using variable-order parametric models with local deformations. *PAMI*, 18(10):972–986, October 1996.

[7] T. Darrell and A. Pentland. Robust estimation of a multi-layered motion representation. In *Workshop on Visual Motion*, pages 173–178, 1991.

[8] T. Darrell and A. Pentland. Cooperative robust estimation using layers of support. *PAMI*, 17(5):474–487, 1995.

[9] B. Glocker, T. H. Heibel, N. Navab, P. Kohli, and C. Rother. Triangleflow: Optical flow with triangulation-based higher-order likelihoods. In *ECCV*, pages 272–285, 2010.

[10] M. Irani, P. Anandan, and D. Weinshall. From reference frames to reference planes: Multi-view parallax geometry and applications. In *ECCV*, 1998.

[11] A. Jepson and M. J. Black. Mixture models for optical flow computation. In *CVPR*, 1993.

[12] N. Jojic and B. Frey. Learning flexible sprites in video layers. In *CVPR*, pages I:199–206, 2001.

[13] A. Kannan, B. Frey, and N. Jojic. A generative model of dense optical flow in layers. Technical Report TR PSI-2001-11, University of Toronto, Aug. 2001.

[14] R. Kumar, P. Anandan, and K. Hanna. Shape recovery from multiple views: A parallax based approach. In *Proc 12th ICPR*, 1994.

[15] R. D. Morris, X. Descombes, and J. Zerubia. The Ising/Potts model is not well suited to segmentation tasks. In *Proceedings of the IEEE Digital Signal Processing Workshop*, 1996.

[16] M. Nicolescu and G. Medioni. Motion segmentation with accurate boundaries - a tensor voting approach. In *CVPR*, pages 382–389, 2003.

[17] M. P. Kumar, P. H. Torr, and A. Zisserman. Learning layered motion segmentations of video. *IJCV*, 76(3):301–319, 2008.

[18] S. Roth and M. J. Black. On the spatial statistics of optical flow. *IJCV*, 74(1):33–50, August 2007.

[19] H. S. Sawhney. 3D geometry from planar parallax. In *CVPR*, pages 929–934, 1994.

[20] T. Schoenemann and D. Cremers. High resolution motion layer decomposition using dual-space graph cuts. In *CVPR*, pages 1–7, June 2008.

[21] E. Sudderth and M. Jordan. Shared segmentation of natural scenes using dependent Pitman-Yor processes. In *NIPS*, pages 1585–1592, 2009.

[22] D. Sun, S. Roth, and M. J. Black. Secrets of optical flow estimation and their principles. In *CVPR*, 2010.

[23] D. Sun, S. Roth, J. P. Lewis, and M. J. Black. Learning optical flow. In *ECCV*, pages 83–97, 2008.

[24] P. Torr, R. Szeliski, and P. Anandan. An integrated Bayesian approach to layer extraction from image sequences. *PAMI*, 23(3):297–303, Mar 2001.

[25] J. Y. A. Wang and E. H. Adelson. Representing moving images with layers. *IEEE Transactions on Image Processing*, 3(5):625–638, Sept. 1994.

[26] Y. Weiss. Smoothness in layers: Motion segmentation using nonparametric mixture estimation. In *CVPR*, pages 520–526, Jun 1997.

[27] Y. Weiss and E. Adelson. A unified mixture framework for motion segmentation: Incorporating spatial coherence and estimating the number of models. In *CVPR*, pages 321–326, Jun 1996.

[28] M. Werlberger, T. Pock, and H. Bischof. Motion estimation with non-local total variation regularization. In *CVPR*, 2010.

[29] H. Yalcin, M. J. Black, and R. Fablet. The dense estimation of motion and appearance in layers. In *IEEE Workshop on Image and Video Registration*, pages 777–784, Jun 2004.

[30] Y. Zhou and H. Tao. Background layer model for object tracking through occlusion. In *ICCV*, volume 2, pages 1079–1085, 2003.

